# Linear-time Algorithms for Pairwise Statistical Problems

**Parikshit Ram, Dongryeol Lee, William B. March and Alexander G. Gray**
Computational Science and Engineering, Georgia Institute of Technology
Atlanta, GA 30332
{p.ram@,dongryel@cc.,march@,agray@cc.}gatech.edu

## Abstract

Several key computational bottlenecks in machine learning involve pairwise distance computations, including all-nearest-neighbors (finding the nearest neighbor(s) for each point, e.g. in manifold learning) and kernel summations (e.g. in kernel density estimation or kernel machines). We consider the general, bichromatic case for these problems, in addition to the scientific problem of N-body simulation. In this paper we show for the first time $\mathbf{O}(N)$ worst case runtimes for practical algorithms for these problems based on the cover tree data structure [1].

## 1 Introduction

Pairwise distance computations are fundamental to many important computations in machine learning and are some of the most expensive for large datasets. In particular, we consider the class of *all-query* problems, in which the combined interactions of a *reference* set $\mathcal{R}$ of $N$ points in $\mathbb{R}^D$ is computed for each point $q$ in a *query* set $\mathcal{Q}$ of size $\mathbf{O}(N)$. This class of problems includes the pairwise kernel summation used in kernel density estimation and kernel machines and the all-nearest neighbors computation for classification and manifold learning. All-query problems can be solved directly by scanning over the $N$ reference points for each of the $\mathbf{O}(N)$ queries, for a total running time of $\mathbf{O}(N^2)$. Since quadratic running times are too slow for even modestly-sized problems, previous work has sought to reduce the number of distance computations needed.

We consider algorithms that employ *space-partitioning trees* to improve the running time. In all the problems considered here, the magnitude of the effect of any reference $r$ on a query $q$ is inversely proportional to the distance metric $d(q, r)$. Therefore, the net effect on the query is dominated by references that are "close by". A space-partitioning tree divides the space containing the point set in a hierarchical fashion, allowing for variable resolution to identify major contributing points efficiently.

**Single-Tree Algorithms.** One approach for employing space-partitioning trees is to consider each query point separately – *i.e.* to consider the all-query problem as many *single-query* problems. This approach lends itself to *single-tree* algorithms, in which the references are stored in a tree, and the tree is traversed once for each query. By considering the distance between the query and a collection of references stored in a tree node, the effect of the references can be approximated or ignored if the distances involved are large enough, with appropriate accuracy guarantees for some methods.

The $kd$-tree structure [2] was developed to obtain the nearest-neighbors of a given query in expected logarithmic time and has also been used for efficient kernel summations [3, 4]. However, these methods lack any guarantees on worst-case running time. A hierarchical data structure was also developed for efficient combined potential calculation in computational physics in Barnes & Hut, 1986 [5]. This data structure provides an $\mathbf{O}(\log N)$ bound on the potential computation for a single query, but has no error guarantees. Under their definition of intrinsic dimension, Karger & Ruhl [6] describe a randomized algorithm with $\mathbf{O}(\log N)$ time per query for nearest neighbor search for low-intrinsic-dimensional data. Krauthgamer & Lee proved their navigating nets algorithm can compute

a single-query nearest-neighbor in $\mathbf{O}(\log N)$ time under a more robust notion of low intrinsic dimensionality. The cover tree data structure [1] improves over these two results by both guaranteeing a worst-case runtime for nearest-neighbor and providing efficient computation in practice relative to $kd$-trees. All of these data structures rely on the triangle inequality of the metric space containing $\mathcal{R}$ in order to *prune* references that have little effect on the query.

**Dual-Tree Algorithms.** The approach described above can be applied to every single query to improve the $\mathbf{O}(N^2)$ running time of all-query problems to $\mathbf{O}(N \log N)$. A faster approach to all-query problems uses an algorithmic framework inspired by efficient particle simulation [7] and generalized to statistical machine learning [8] which takes advantage of spatial proximity in both $\mathcal{Q}$ and $\mathcal{R}$ by constructing a space-partitioning tree on each set. Both trees are descended, allowing the contribution from a distant reference node to be pruned for an entire node of query points. These *dual-tree algorithms* have been shown to be significantly more efficient in practice than the corresponding single-tree algorithms for nearest neighbor search and kernel summations [9, 10, 11]. Though conjectured to have $\mathbf{O}(N)$ growth, they lack rigorous, general runtime bounds.

All-query problems fall into two categories: *monochromatic*, where $\mathcal{Q} = \mathcal{R}$ and *bichromatic*, where $\mathcal{Q}$ is distinct from $\mathcal{R}$. Most of the existing work has only addressed the monochromatic case. The fast multipole method (FMM)[7] for particle simulations, considered one of the breakthrough algorithms of the 20th century, has a non-rigorous runtime analysis based on the uniform distribution. An improvement to the FMM for the $N$-body problem was suggested by Aluru,et.al. [12], but was regarding the construction time of the tree and not the querying time. Methods based on the well-separated pair decomposition (WSPD) [13] have been proposed for the all nearest neighbors problem and particle simulations [14], but are inefficient in practice. These methods have $\mathbf{O}(N)$ runtime bounds for the monochromatic case, but it is not clear how to extend the analysis to a bichromatic problem. In addition to this difficulty, the WSPD-based particle simulation method is restricted to the $(1/r)$-kernel. In Beygelzimer et.al., 2006 [1], the authors conjecture, but do not prove, that the cover tree data structure using a dual-tree algorithm can compute the monochromatic all-nearest-neighbors problem in $\mathbf{O}(N)$.

**Our Contribution.** In this paper, we prove $\mathbf{O}(N)$ runtime bounds for several important instances of the dual-tree algorithms for the first time using the cover tree data structure [1]. We prove the first worst-case bounds for any practical kernel summation algorithms. We also provide the first general runtime proofs for dual-tree algorithms on bichromatic problems. In particular, we give the first proofs of worst-case $\mathbf{O}(N)$ runtimes for the following all-query problems:

- **All Nearest-neighbors:** For all queries $q \in \mathcal{Q}$, find $r^*(q) \in \mathcal{R}$ such that $r^*(q) = \arg\min_{r \in \mathcal{R}} d(q, r)$.
- **Kernel summations:** For a given kernel function $K(\cdot)$, compute the kernel summation $f(q) = \sum_{r \in \mathcal{R}} K(d(q, r))$ for all $q \in \mathcal{Q}$.
- **N-body potential calculation:** Compute the net electrostatic or gravitational potential $f(q) = \sum_{r \in \mathcal{R}, r \neq q} d(q, r)^{-1}$ at each $q \in \mathcal{Q}$.

**Outline.** In the remainder of this paper, we give our linear running time proofs for dual-tree algorithms. In Section 2, we review the cover tree data structure and state the lemmas necessary for the remainder of the paper. In Section 3, we state the dual-tree all-nearest-neighbors algorithm and prove that it requires $\mathbf{O}(N)$ time. In Section 4, we state the absolute and relative error guarantees for kernel summations and again prove the linear running time of the proposed algorithms. In the same section, we apply the kernel summation result to the $N$-body simulation problem from computational physics, and we draw some conclusions in Section 5.

## 2    Cover Trees

A cover tree [1] $T$ stores a data set $\mathcal{R}$ of size $N$ in the form of a levelled tree. The structure has an $\mathbf{O}(N)$ space requirement and $\mathbf{O}(N \log N)$ construction time. Each level is a "cover" for the level beneath it and is indexed by an integer scale $i$ which decreases as the tree is descended. Let $C_i$ denote the set of nodes at scale $i$. For all scales $i$, the following invariants hold:

- (nesting invariant) $C_i \subset C_{i-1}$
- (covering tree invariant) For every $p \in C_{i-1}$, there exists a $q \in C_i$ satisfying $d(p, q) \leq 2^i$, and exactly one such $q$ is a parent of $p$.
- (separation invariant) For all $p, q \in C_i$, $d(p, q) > 2^i$.

**Representations.**  The cover tree has two different representations: The *implicit representation* consists of infinitely many levels $C_i$ with the level $C_\infty$ containing a single node which is the root and the level $C_{-\infty}$ containing every point in the dataset as a node. The *explicit representation* is required to store the tree in $\mathbf{O}(N)$ space. It coalesces all nodes in the tree for which the only child is the self-child. This implies that every explicit node either has a parent other than the self-parent or has a child other than a self-child.

**Structural properties.**  The intrinsic dimensionality measure considered here is the *expansion dimension* from Karger & Ruhl, 2002 [6] defined as follows:

**Definition 2.1.** *Let $B_\mathcal{R}(p, \rho) = \{r \in \mathcal{R} \subset X \colon d(p, r) \leq \rho\}$ denote a closed ball of radius $\rho$ around a $p \in \mathcal{R}$. Then, the **expansion constant** of $\mathcal{R}$ is defined as the smallest $c \geq 2$ such $|B_\mathcal{R}(p, 2\rho)| \leq c|B_\mathcal{R}(p, \rho)| \ \forall p \in \mathcal{R}$ and $\forall \rho > 0$. The intrinsic dimensionality (or expansion dimension) of $\mathcal{R}$ is given by $d_{KR}(\mathcal{R}) = \log c$.*

We make use of the following lemmas from Beygelzimer et.al., 2006 [1] in our runtime proofs.

**Lemma 2.1.** *(Width bound) The number of children of any node $p$ is bounded by $c^4$.*

**Lemma 2.2.** *(Growth bound) For all $p \in \mathcal{R}$ and $\rho > 0$, if there exists a point $r \in \mathcal{R}$ such that $2\rho < d(p, r) \leq 3\rho$, then $|B(p, 4\rho)| \geq \left(1 + \frac{1}{c^2}\right)|B(p, \rho)|$.*

**Lemma 2.3.** *(Depth bound) The maximum depth of any point $p$ in the explicit representation is $\mathbf{O}(c^2 \log N)$.*

**Single point search: Single tree nearest neighbor.**  Given a cover tree $T$ built on a set $\mathcal{R}$, the nearest neighbor of a query $q$ can be found with the **FindNN** subroutine in Algorithm 1. The algorithm uses the triangular inequality to prune away portions of the tree that contain points distant from $q$. The following theorem provides a runtime bound for the single point search.

**Theorem 2.1.** *(Query time) If the dataset $\mathcal{R} \cup \{q\}$ has expansion constant $c$, the nearest neighbor of $q$ can be found in time $\mathbf{O}(c^{12} \log N)$.*

**Batch Query:** The dual tree algorithm for all-nearest-neighbor (**FindAllNN** subroutine in Algorithm 1) using cover trees is provided in Beygelzimer et.al., 2006 [15] as batch-nearest-neighbor.

# 3  Runtime Analysis of All-Nearest-Neighbors

In the bichromatic case, the performance of the **FindAllNN** algorithm (or any dual-tree algorithm) will depend on the degree of difference between the query and reference sets. If the sets are nearly identical, then the runtime will be close to the monochromatic case. If the inter-point distances in the query set are very large relative to those between references, then the algorithm may have to descend to the leaves of the query tree before making any descends in the reference tree. This case offers no improvement over the performance of the single-tree algorithm applied to each query. In order to quantify this difference in scale for our runtime analysis, we introduce the *degree of bichromaticity*:

**Definition 3.1.** *Let $S$ and $T$ be cover trees built on query set $\mathcal{Q}$ and reference set $\mathcal{R}$ respectively. Consider a dual-tree algorithm with the property that the scales of $S$ and $T$ are kept as close as possible – i.e. the tree with the larger scale is always descended. Then, the **degree of bichromaticity** $\kappa$ of the query-reference pair $(\mathcal{Q}, \mathcal{R})$ is the maximum number of descends in $S$ between any two descends in $T$.*

In the monochromatic case, the trees are identical and the traversal alternates between them. Thus, the degree of bichromaticity is $\kappa = 1$. As the difference in scales of the two data sets increases, more descends in the query tree become necessary, giving a higher degree of bichromaticity. Using this definition, we can prove the main result of this section.

**Theorem 3.1.** *Given a reference set $\mathcal{R}$ of size $N$ and expansion constant $c_\mathcal{R}$, a query set $\mathcal{Q}$ of size $\mathbf{O}(N)$ and expansion constant $c_\mathcal{Q}$, and bounded degree of bichromaticity $\kappa$ of the $(\mathcal{Q}, \mathcal{R})$ pair, the **FindAllNN** subroutine of Algorithm 1 computes the nearest neighbor in $\mathcal{R}$ of each point in $\mathcal{Q}$ in $\mathbf{O}(c_\mathcal{R}^{12} c_\mathcal{Q}^{4\kappa} N)$ time.*

*Proof.* The computation at Line 3 is done for each of the query nodes at most once, hence takes $\mathbf{O}(\max_i |R_i| * N)$ computations.

The traversal of a reference node is duplicated over the set of queries only if the query tree is descended just before the reference tree descend. For every query descend, there would be at most $\mathbf{O}(c_\mathcal{Q}^4)$ duplications (width bound) for every reference node traversal. Since the number of query

**Algorithm 1** Single tree and batch query algorithm for Nearest Neighbor search and Approximate Kernel summation

---

**FindNN($\mathcal{R}$-Tree $T$, query $q$)**

    **Initialize** $R_\infty = C_\infty$.
    **for** $i = \infty$ **to** $-\infty$ **do**
3:    $R = \{Children(r) : r \in R_i\}$
      $R_{i-1} = \{r \in R : d(q,r) \leq d(q,R) + 2^i\}$
    **end for**
6: **return** $\arg\min_{r \in R_{-\infty}} d(q,r)$

**FindAllNN($\mathcal{Q}$-subtree $q_j$, $\mathcal{R}$-cover set $R_i$)**
    **if** $i = -\infty$ **then**
      $\forall q \in L(q_j)$ **return** $\arg\min_{r \in R_{-\infty}} d(q,r)$.
      // $L(q_j)$ is the set of all the leaves of the subtree $q_j$.
3: **else if** $j < i$ **then**
      $R = \{Children(r) : r \in R_i\}$
      $R_{i-1} = \{r \in R :$
           $d(q_j,r) \leq d(q_j,R) + 2^i + 2^{j+2}\}$
6:    **FindAllNN** $(q_j, R_{i-1})$
    **else**
      $\forall p_{j-1} \in Children(q_j)$ **FindAllNN**$(p_{j-1}, R_i)$
9: **end if**

**KernelSum($\mathcal{R}$-tree $T$, query $q$)**

    **Initialize** $R_\infty = C_\infty$, $\hat{f}(q) = 0$
    **for** $i = \infty$ **to** $-\infty$ **do**
3:    $R = \{Children(r) : r \in R_i\}$
      $R_{i-1} = \{r \in R : K_h(d(q,r) - 2^i)$
              $-K_h(d(q,r) + 2^i) > \epsilon\}$
      $\hat{f}(q) = \hat{f}(q) + \sum_{r \in \{R - R_{i-1}\}} K_h(d(q,r)) \cdot |L(r)|$
6: **end for**
    **return** $\hat{f}(q) = \hat{f}(q) + \sum_{r \in R_{-\infty}} K_h(d(q,r))$

**Initialize** $\Delta_f(q) \leftarrow 0 \forall q \in q_\infty$
**AllKernelSum($\mathcal{Q}$-subtree $q_j$,**
          **$\mathcal{R}$-cover set $R_i$)**
    **if** $i = -\infty$ **then**
      **for** $\forall q \in L(q_j)$ **do**
3:      $\hat{f}(q) = \hat{f}(q)$
             $+ \sum_{r \in R_{-\infty}} K_h(d(q,r))$
             $+\Delta_f(q_j)$
      **end for**
      $\Delta_f(q_j) = 0$
6: **else**
      **if** $j < i$ **then**
        $R = \{Children(r) : r \in R_i\}$
9:      $R_{i-1} = \{r \in R :$
        $K_h(d(q_j,r) - 2^i - 2^{j+1})$
        $-K_h(d(q_j,r) + 2^i + 2^{j+1})$
        $> \epsilon\}$
        $\Delta_f(q_j) = \Delta_f(q_j) +$
           $\sum_{r \in R \setminus R_{i-1}} K_h(d(q_j,r)) \cdot |L(r)|$
        **AllKernelSum**$(q_j, R_{i-1})$
12:     **else**
        **for** $\forall p_{j-1} \in Children(q_j)$ **do**
          $\Delta_f(p_{j-1}) = \Delta_f(p_{j-1}) + \Delta_f(q_j)$
15:      **AllKernelSum**$(p_{j-1}, R_i)$
        **end for**
        $\Delta_f(q_j) = 0$
18:    **end if**
    **end if**

---

descends between any two reference descends is upper bounded by $\kappa$ and the number of explicit reference nodes is $\mathbf{O}(N)$, the total number of reference node considered in Line 5 in the whole algorithm is at most $\mathbf{O}(c_\mathcal{Q}^{4\kappa} N)$.

Since at any level of recursion, the size of $R$ is bounded by $c_\mathcal{R}^4 \max_i |R_i|$ (width bound), and the maximum depth of any point in the explicit tree is $\mathbf{O}(c_\mathcal{R}^2 \log N)$ (depth bound), the number of nodes encountered in Line 6 is $\mathbf{O}(c_\mathcal{R}^{4+2} \max_i |R_i| \log N)$. Since the traversal down the query tree causes duplication, and the duplication of any reference node is upper bounded by $c_\mathcal{Q}^{4\kappa}$, Line 6 takes at most $\mathbf{O}(c_\mathcal{Q}^{4\kappa} c_\mathcal{R}^6 \max_i |R_i| \log N)$ in the whole algorithm.

Line 9 is executed just once for each of the explicit nodes of the query tree and hence takes at most $\mathbf{O}(N)$ time.

Consider any $R_{i-1} = \{r \in R : d(q_j,r) \leq d + 2^i + 2^{j+2}\}$ where $d = d(q_j, R)$. Given that $C_{i-1}$ is the $(i-1)^{th}$ level of the reference tree $R_{i-1} = B(q_j, d + 2^i + 2^{j+2}) \cap R \subseteq B(q_j, d + 2^i + 2^{j+2}) \cap C_{i-1} \subseteq B(q_j, d + 2^i + 2^{i+1}) \cap C_{i-1}$ since $R \subseteq C_{i-1}$ and $j < i$ in this part of the recursion. If $d > 2^{i+2}$, $|B(q_j, d + 2^i + 2^{i+1})| \leq |B(q_j, 2d)| \leq c_\mathcal{R}^2 |B(q_j, \frac{d}{2})|$. Now $d \leq d(q_j, \mathcal{R}) + 2^i$ since $R \subseteq C_{i-1}$ and $d > 2^{i+2}$, $d(q_j, \mathcal{R}) > 2^{i+1}$, making $|B(q_j, \frac{d}{2})| = |\{q_j\}| = 1$. Hence $|R_{i-1}| \leq c_\mathcal{R}^2$.

If $d \leq 2^{i+2}$, as in Beygelzimer et.al. [1] the number of disjoint balls of radius $2^{i-2}$ that can be packed in $B(q_j, d + 2^i + 2^{i+1})$ is bounded as $|B(q_j, d + 2^i + 2^{i+1} + 2^{i-2})| \leq |B(r, 2(d + 2^i + 2^{i+1}) + 2^{i-2})| \leq |B(r, 2^{i+3} + 2^{i+1} + 2^{i+2} + 2^{i-2})| \leq |B(r, 2^{i+4})| \leq |c_\mathcal{R}^6 B(r, 2^{i-2})|$ for some $r \in C_{i-1}$. Any such ball $B(r, 2^{i-2})$ can contain at most one point in $C_{i-1}$, making $|R_{i-1}| \leq c_\mathcal{R}^6$.

Thus, the algorithm takes $\mathbf{O}(c_{\mathcal{R}}^6 N + c_{\mathcal{Q}}^{4\kappa} N + c_{\mathcal{R}}^{12} c_{\mathcal{Q}}^{4\kappa} \log N + N)$ which is $\mathbf{O}(c_{\mathcal{R}}^{12} c_{\mathcal{Q}}^{4\kappa} N)$. $\qquad\square$

**Corollary 3.1.** *In the monochromatic case with a dataset $\mathcal{R}$ of size $N$ having an expansion constant c, the **FindAllNN** subroutine of Algorithm 1 has a runtime bound of $\mathbf{O}(c^{16} N)$.*

*Proof.* In the monochromatic case, $|\mathcal{Q}| = |\mathcal{R}| = N$, $c_{\mathcal{Q}} = c_{\mathcal{R}} = c$ and the degree of bichromaticity $\kappa = 1$ since the query and the reference tree are the same. Therefore, by Theorem 3.1, the result follows. $\qquad\square$

## 4   Runtime Analysis of Approximate Kernel Summations

For infinite tailed kernels $K(\cdot)$, the exact computation of kernel summations is infeasible without $\mathbf{O}(N^2)$ operations. Hence the goal is to efficiently approximate $f(q) = \sum_r K(d(q,r))$ where $K(\cdot)$ is a monotonically decreasing non-negative kernel function. We employ the two widely used approximating schemes listed below:

**Definition 4.1.** *An algorithm guarantees $\epsilon$ **absolute error bound**, if for each exact value $f(q_i)$ for $q_i \in \mathcal{Q}$, it computes $\hat{f}(q_i)$ such that $\left| \hat{f}(q_i) - f(q_i) \right| \leq N\epsilon$.*

**Definition 4.2.** *An algorithm guarantees $\epsilon$ **relative error bound**, if for each exact value $f(q_i)$ for $q_i \in \mathcal{Q}$, it computes $\hat{f}(q_i) \in \mathbb{R}$ such that $\left| \hat{f}(q_i) - f(q_i) \right| \leq \epsilon |f(q_i)|$.*

Approximate kernel summation is more computationally intensive than nearest neighbors because pruning is not based on the distances alone but also on the analytical properties of the kernel (*i.e.* smoothness and extent). Therefore, we require a more extensive runtime analysis, especially for kernels with an infinite extent, such as the Gaussian kernel. We first prove logarithmic running time for the single-query kernel sum problem under an absolute error bound and then show linear running time for the dual-tree algorithm. We then extend this analysis to include relative error bounds.

### 4.1   Single Tree Approximate Kernel Summations Under Absolute Error

The algorithm for computing the approximate kernel summation under absolute error is shown in the **KernelSum** subroutine of Algorithm 1. The following theorem proves that **KernelSum** produces an approximation satisfying the $\epsilon$ absolute error.

**Theorem 4.1.** *The **KernelSum** subroutine of Algorithm 1 outputs $\hat{f}(q)$ such that $|\hat{f}(q) - f(q)| \leq N\epsilon$.*

*Proof.* A subtree rooted at $r \in C_{i-1}$ is pruned as per Line 5 of **KernelSum** since for $\forall r' \in L(r)$, $K(d(q,r) + 2^i) \leq K(d(q,r')) \leq K(d(q,r) - 2^i)$ and $|K(d(q,r)) - K(d(q,r'))| \leq \epsilon$. This amounts to limiting the error per each kernel evaluation to be less than $\epsilon$ (which also holds true for each contribution computed exactly for $r \in R_{-\infty}$, and by the triangle inequality the kernel approximate sum $\hat{f}(q)$ will be within $N\epsilon$ of the true kernel sum $f(q)$. $\qquad\square$

The following theorem proves the runtime of the single-query kernel summation with smooth and monotonically decreasing kernels using a cover tree.

**Theorem 4.2.** *Given a reference set $\mathcal{R}$ of size $N$ and expansion constant c, an error value $\epsilon$, and a monotonically decreasing smooth non-negative kernel function $K(\cdot)$ concave for $x \in [0, h]$ and convex for $x \in (h, \infty)$ for some $h > 0$, the **KernelSum** subroutine of Algorithm 1 computes the kernel summation at a query q approximately up to $\epsilon$ absolute error with a runtime bound of $\mathbf{O}(c^{2(1 + \max\{\eta - i_1 + 3, \gamma - i_1 + 4, 4\})} \log N)$ time where*
$\eta = \lceil \log_2 K^{(-1)}(\epsilon) \rceil$, $\gamma = \lceil \log_2 h \rceil$, $i_1 = \left\lfloor \log_2 \left( \frac{-\epsilon}{K'(h)} \right) \right\rfloor$, *and $K'(\cdot)$ is the derivative of $K(\cdot)$.*

*Proof.* We assume that any argument of $K(\cdot)$ is lower bounded at 0. Now define the following sets:

$$R_{i-1}^l = \{r \in R_{i-1} : d(q,r) \leq h - 2^i\}$$
$$R_{i-1}^m = \{r \in R_{i-1} : h - 2^i < d(q,r) \leq h + 2^i\}$$
$$R_{i-1}^u = \{r \in R_{i-1} : d(q,r) > h + 2^i\}$$

such that $R_{i-1} = R_{i-1}^l \cup R_{i-1}^m \cup R_{i-1}^u$, and are pairwise disjoint. For $r \in R_{i-1}^l$:

$$\epsilon < K(\max(0, (d(q,r) - 2^i))) - K(d(q,r) + 2^i)$$
$$\leq (K(d(q,r) + 2^i) - 2^{i+1} K'(d(q,r) + 2^i)) - K(d(q,r) + 2^i) = -2^{i+1} K'(d(q,r) + 2^i)$$

because of the concavity of the kernel function $K(\cdot)$. Now,

$$K'^{(-1)}_{[0,h-2^i]}\left(\frac{-\epsilon}{2^{i+1}}\right) - 2^i < d(q,r) \le h - 2^i$$

where $K'^{(-1)}_{[a,b]}(x)$ is 1) the inverse function of the $K'(x)$; 2) the output value is restricted to be in the interval $[a,b]$ for the given argument $x$. For $r \in R^m_{i-1}$,

$$\epsilon < K(\max(0,(d(q,r)-2^i))) - K(d(q,r)+2^i) \le -2^{i+1}K'(h)$$

which implies that

$$i \ge \log_2\left(\frac{-\epsilon}{K'(h)}\right) - 1$$

Similarly, for $r \in R^u_{i-1}$, $\epsilon < -2^{i+1}K'(d(q,r) - 2^i)$ implying

$$h + 2^i < d(q,r) < K'^{(-1)}_{(h+2^i,\infty)}\left(\frac{-\epsilon}{2^{i+1}}\right) + 2^i.$$

Note that $0 \ge K'(d(q,r)) \ge K'(h)$ for $d(q,r) > h + 2^i$, which implies that $\frac{-\epsilon}{2^{i+1}} \ge K'(h)$ and thus $i \ge \left\lfloor \log_2\left(\frac{-\epsilon}{K'(h)}\right)\right\rfloor = i_1$. Below the level $i_1$, $R^l_{i-1} = R^u_{i-1} = \emptyset$. In addition, below the level $i_1 - 1$, $R^m_{i-1} = \emptyset$.

Case 1: $i > i_1$
Trivially, for $r \in R_{i-1}$, $K(d^{max} - 2^i) > \epsilon$ where $d^{max} = \max_{r \in R_{i-1}} d(q,r)$. We can invert the kernel function to obtain: $d^{max} < K^{(-1)}_{(h+2^i,\infty)}(\epsilon) + 2^i$. This implies that $d(q,r) \le d^{max} < K^{(-1)}(\epsilon) + 2^i$ We can count up the number of balls of radius $2^{i-2}$ inside $B\left(q, K^{(-1)}(\epsilon) + 2^i + 2^{i-2}\right)$. Let $\eta = \left\lceil \log_2 K^{(-1)}(\epsilon)\right\rceil$. Then,

$$\max|R_{i-1}| \le |B(q, 2^\eta + 2^i + 2^{i-2}) \cap C_{i-1}| \le \begin{cases} |B(q, 2^{i+1}) \cap C_{i-1}| \le c^3, \eta < i \\ |B(q, 2^{i+2}) \cap C_{i-1}| \le c^4, \eta = i \\ |B(q, 2^{\eta+1}) \cap C_{i-1}| \le c^{\eta-i+3} = c^{\eta-i_1+3}, \eta > i \end{cases}$$

Case 2: $i = i_1 - 1$
Let $\gamma = \lceil \log_2 h \rceil$. Similar to the case above, we count the number of balls of radius $2^{i-2}$ inside $B\left(q, 2^\gamma + 2^i + 2^{i-2}\right)$.

$$\max|R_{i-1}| \le |B(q, 2^\gamma + 2^i + 2^{i-2}) \cap C_{i-1}| \le \begin{cases} |B(q, 2^{i+1}) \cap C_{i-1}| \le c^3, \gamma < i \\ |B(q, 2^{i+2}) \cap C_{i-1}| \le c^4, \gamma = i \\ |B(q, 2^{\gamma+1}) \cap C_{i-1}| \le c^{\gamma-i+3} = c^{\gamma-i_1+4}, \gamma > i \end{cases}$$

From the runtime proof of the single-tree nearest neighbor algorithm using cover tree in Beygelzimer et.al., 2006, the running time is bounded by:

$$\mathbf{O}(k\max|R_{i-1}|^2 + k\max|R_{i-1}|c^4) \le \mathbf{O}(c^{2(1+\max\{\eta-i_1+3,\gamma-i_1+4,4\})}\log N)$$

$\square$

## 4.2 Dual Tree Approximate Kernel Summations Under Absolute Error

An algorithm for the computation of kernel sums for multiple queries is shown in the **AllKernelSum** subroutine of Algorithm 1, analogous to **FindAllNN** for batch nearest-neighbor query. The dual-tree version of the algorithm requires a stricter pruning rule to ensure correctness for all the queries in a query subtree. Additionally, every query node $q_j$ has an associated $\mathbf{O}(1)$ storage $\Delta_f(q_j)$ that accumulates the *postponed* kernel contribution for all query points under the subtree $q_j$. The following theorem proves the correctness of the **AllKernelSum** subroutine of Algorithm 1.

**Theorem 4.3.** *For all $q$ in the in the query set $\mathcal{Q}$, the **AllKernelSum** subroutine of Algorithm 1 computes approximations $\hat{f}(q)$ such that $|\hat{f}(q) - f(q)| \le N\epsilon$.*

*Proof.* Line 9 of the algorithm guarantees that $\forall r \in R \backslash R_{i-1}$ at a given level $i$,

$$|K(d(q_j,r)) - K(d(q,r))| \le |K(d(q_j,r) - 2^i - 2^{j+1}) - K(d(q_j,r) + 2^i + 2^{j+1})| \le \epsilon$$

for all $q \in L(q_j)$. Basically, the minimum distance is decreased and the maximum distance is increased by $2^{j+1}$, which denotes the maximum possible distance from $q_j$ to any of its descendants. Trivially, contributions added in Line 3 (the base case) satisfy the $\epsilon$ absolute error for each kernel value and the result follows by the triangle inequality. $\square$

Based on the runtime analysis of the batch nearest neighbor, the runtime bound of **AllKernelSum** is given by the following theorem:

**Theorem 4.4.** *Let $\mathcal{R}$ be a reference set of size $N$ and expansion constant $c_{\mathcal{R}}$, and let $\mathcal{Q}$ be a query set of size $\mathbf{O}(N)$ and expansion constant $c_{\mathcal{Q}}$. Let the $(\mathcal{Q}, \mathcal{R})$ pair have a bounded degree of bichromaticity. Let $K(\cdot)$ be a monotonically-decreasing smooth non-negative kernel function that is concave for $x \in [0, h]$ and convex for $x \in (h, \infty)$ for some $h > 0$. Then, given an error tolerance $\epsilon$, the **AllKernelSum** subroutine of Algorithm 1 computes an approximation $\hat{f}(q)\ \forall q \in \mathcal{Q}$ that satisfies the $\epsilon$ absolute error bound in time $\mathbf{O}(N)$.*

*Proof.* We first bound $\max |R_{i-1}|$. Note that in Line 9 to Line 13 of the **AllKernelSum**, $j \leq i + 1$, and thus $2^i + 2^{j+1} \leq 2^i + 2^i = 2^{i+1}$. Similar to the proof for the single-tree case, we define:

$$R_{i-1}^l = \{r \in R_{i-1} : d(q, r) \leq h - 2^{i+1}\}$$
$$R_{i-1}^m = \{r \in R_{i-1} : h - 2^{i+1} < d(q, r) \leq h + 2^{i+1}\}$$
$$R_{i-1}^u = \{r \in R_{i-1} : d(q, r) > h + 2^{i+1}\}$$

such that $R_{i-1} = R_{i-1}^l \cup R_{i-1}^m \cup R_{i-1}^u$, and pairwise disjoint. From here, we can follow the techniques shown for the single-tree case to show that $\max |R_{i-1}|$ is constant dependent on $c$. Therefore, the methodology of the runtime analysis of batch nearest neighbor gives the $\mathbf{O}(N)$ runtime for batch approximate kernel summation. $\qquad\square$

## 4.3 Approximations Under Relative Error

We now extend the analysis for absolute error bounds to cover approximations under the relative error criterion given in Definition 4.2.

**Single-tree case.** For a query point $q$, the goal is compute $\hat{f}(q)$ satisfying Definition 4.2. An approximation algorithm for a relative error bound is similar to the **KernelSum** subroutine of Algorithm 1 except that the definition of $R_{i-1}$ (*i.e.* the set of reference points that are not pruned at the given level $i$) needs to be changed to satisfy the relative error constraint as follows:

$$R_{i-1} = \{r \in R : K(d(q, r) - 2^i) - K(d(q, r) + 2^i) > \frac{\epsilon f(q)}{N}\}$$

where $f(q)$ is the unknown query sum. Hence, let $d^{max} = \max_{r \in \mathcal{R}} d(q, r)$, and expand the set $R_{i-1}$ to:

$$R_{i-1} \subseteq \{r \in R : K(d(q, r) - 2^i) - K(d(q, r) + 2^i) > \epsilon K(d^{max})\} \tag{1}$$

Note that $d^{max}$ can be trivially upper bounded by: $d^{max} \leq d(q, r_{root}) + 2^{p+1} = d^{max,u}$ where $p$ is the scale of the root of the reference cover tree in the explicit representation.

**Theorem 4.5.** *Let the conditions of Thm. 4.2 hold. Then, the **KernelSum** subroutine of Algorithm 1 with Line 5 redefined as Eqn. 1 computes the kernel summation $\hat{f}(q)$ at a query $q$ with $\epsilon$ relative error in $\mathbf{O}(\log N)$ time.*

*Proof.* A node $r \in C_{i-1}$ can be pruned by the above pruning rule since for $r' \in L(r)$, $K(d(q, r) + 2^i) \leq K(d(q, r')) \leq K(d(q, r) - 2^i)$ and $|K(d(q, r)) - K(d(q, r'))| \leq \epsilon K(d^{max,u})$. This amounts to limiting the error per each kernel evaluation to be less than $\epsilon K(d^{max,u})$ (which also holds true for each contribution computed exactly for $r \in R_{-\infty}$, and by the triangle inequality the kernel approximate sum $\hat{f}(q)$ will be within $\epsilon N K(d^{max,u}) \leq \epsilon f(q)$ of the true kernel sum $f(q)$. Since the relative error is an instance of the absolute error, the algorithm also runs in $\mathbf{O}(\log N)$. $\qquad\square$

**Dual-tree case.** In this case, for each query point $q \in \mathcal{Q}$, an approximation $\hat{f}(q)$ is to be computed as per Definition 4.2. As in the absolute error case, we must satisfy a more difficult condition. Therefore, $d^{max,u}$ is larger, taking into account both the maximum possible distance from the root of the query tree to its descendants and the maximum possible distance from the root of the reference tree to its descendants. Hence $R_{i-1}$ is defined as follows:

$$R_{i-1} = \{r \in R : K(d(q, r) - 2^i - 2^{j+1}) - K(d(q, r) + 2^i + 2^{j+1}) > \epsilon K(d^{max,u})\} \tag{2}$$

where $d(q_{root}, r_{root}) + 2^{p_{\mathcal{Q}}+1} + 2^{p_{\mathcal{R}}+1} = d^{max,u}$ and $p_{\mathcal{Q}}, p_{\mathcal{R}}$ are the scales of the roots of the query and reference cover trees respectively in the explicit representations. The correctness of the algorithm follows naturally from Theorems 4.4 and 4.5.

**Corollary 4.1.** *Let the conditions of Thm. 4.4 hold. Then, given an error value $\epsilon$, the **AllKernelSum** subroutine of Algorithm 1 with Line 11 redefined as Eq. 2 computes an approximate kernel summation $\hat{f}(q)\ \forall q \in \mathcal{Q}$ that satisfies an $\epsilon$ relative error bound with a runtime bound of $\mathbf{O}(N)$.*

Note that for the single-tree and dual-tree algorithms under the relative error criterion, the pruning rules that generate $R_{i-1}$ shown above are sub-optimal in practice, because they require every pairwise kernel value that is pruned to be within $\epsilon$ relative error. There is a more sophisticated way of accelerating this using an alternative method [9, 10, 11] that is preferable in practice.

### 4.4 $N$-body Simulation

$N$-body potential summation is an instance of the kernel summation problem that arises in computational physics and chemistry. These computations use the Coulombic kernel $K(d) = 1/d$, which describes gravitational and electrostatic interactions. This kernel is infinite at zero distance and has no inflection point (*i.e.* it is convex for $d \in (0, \infty)$). Nevertheless, it is possible to obtain the runtime behavior using the results shown in the previous sections. The single query problem $f(q) = \sum_r \frac{1}{d(q,r)}$ is considered first under the assumption that $\min_{r \in \mathcal{R}, q \neq r} d(q, r) > 0$.

**Corollary 4.2.** *Given a reference set $\mathcal{R}$ of size $N$ and expansion constant $c$, an error value $\epsilon$ and the kernel $K(d) = 1/d(q,r)$, the **KernelSum** subroutine of Algorithm 1 computes the potential summation at a query $q$ with $\epsilon$ error in $\mathbf{O}(\log N)$ time.*

*Proof.* Let $d^{min} = \min\limits_{r \in \mathcal{R}, q \neq r} d(q,r)$. Let $K^e(d)$ be the $C^2$ continuous construction [16] such that:

$$K_e(d) = \begin{cases} \frac{1}{d^{min}}\left(\frac{15}{8} - \frac{5}{4}\left(\frac{d}{d^{min}}\right)^2 + \frac{3}{8}\left(\frac{d}{d^{min}}\right)^4\right), d < d^{min} \\ \frac{1}{d}, d \geq d^{min} \end{cases}$$

The effective kernel $K_e(d)$ can be constructed in $\mathbf{O}(\log N)$ time using the single-tree algorithm for nearest neighbor described in Beygelzimer et.al., 2006 [1]. Note that the second derivative of the effective kernel is $K_e''(d) = \frac{-5}{2(d^{min})^3} + \frac{9d^2}{2(d^{min})^5}$ for $d < d^{min}$. Thus it is concave for $d < \frac{\sqrt{5}}{3}d^{min}$ and convex otherwise, so the second derivative agrees at $d = d^{min}$. Note that $K_e(d)$ agrees with $K(d)$ for $d \geq d^{min}$. Hence, by considering $d^{min}$ equivalent to the bandwidth $h$ in Theorem 4.2 and applying the same theorem on the **KernelSum** subroutine of Algorithm 1 with the aforementioned kernel, we prove the $\mathbf{O}(\log N)$ runtime bound. $\qquad\square$

The runtime analysis for the batch case of the algorithm follows naturally.

**Corollary 4.3.** *Given a reference set $\mathcal{R}$ of size $N$ and expansion constant $c_{\mathcal{R}}$ and a query set $\mathcal{Q}$ of size $\mathbf{O}(N)$ and expansion constant $c_{\mathcal{Q}}$ with a bounded degree of bichromaticity for the $(\mathcal{Q}, \mathcal{R})$ pair, an error value $\epsilon$ and the kernel $K(d) = 1/d(q,r)$, the **AllKernelSum** subroutine of Algorithm 1 approximates the potential summation $\forall q \in \mathcal{Q}$ up to $\epsilon$ error with a runtime bound of $\mathbf{O}(N)$.*

*Proof.* The same effective kernel as Corollary 4.2 is used, except that $d^{min} = \min\limits_{q \in \mathcal{Q}} \min\limits_{r \in \mathcal{R}, q \neq r} d(q,r)$.
The result follows from applying Theorem 4.4, and noting that running the dual-tree computation with $K(d(q,r)) = 1/d(q,r)$ is equivalent to running the algorithm with $K_e(d(q,r))$. $\qquad\square$

## 5 Conclusions

Extensive work has attempted to reduce the quadratic scaling of the all-query problems in statistical machine learning. So far, the improvements in runtimes have only been empirical with no rigorous runtime bounds [2, 8, 9, 17, 18]. Previous work has provided algorithms with rough linear runtime arguments for certain instances of these problems [14, 5, 13], but these results only apply to the monochromatic case. In this paper, we extend the existing work [6, 1, 19, 20] to provide algorithms for two important instances of the all-query problem (namely all-nearest-neighbor and all-kernel-summation) and obtain for the first time a linear runtime bound for dual-tree algorithms for the more general bichromatic case of the all-query problems.

These results provide an answer to the long-standing question of the level of improvement possible over the quadratic scaling of the all-query problems. The techniques used here finally point the way to analyzing a host of other tree-based algorithms used in machine learning, including those that involve $n$-tuples, such as the $n$-point correlation (which naïvely require $\mathbf{O}(N^n)$ computations).

# References

[1] A. Beygelzimer, S. Kakade, and J.C. Langford. Cover Trees for Nearest Neighbor. *Proceedings of the 23rd International Conference on Machine learning*, pages 97–104, 2006.

[2] J. H. Freidman, J. L. Bentley, and R. A. Finkel. An Algorithm for Finding Best Matches in Logarithmic Expected Time. *ACM Trans. Math. Softw.*, 3(3):209–226, September 1977.

[3] K. Deng and A. W. Moore. Multiresolution Instance-Based Learning. pages 1233–1242.

[4] D. Lee and A. G. Gray. Faster Gaussian Summation: Theory and Experiment. In *Proceedings of the Twenty-second Conference on Uncertainty in Artificial Intelligence*. 2006.

[5] J. Barnes and P. Hut. A Hierarchical $O(N \log N)$ Force-Calculation Algorithm. *Nature*, 324, 1986.

[6] D. R. Karger and M. Ruhl. Finding Nearest Neighbors in Growth-Restricted Metrics. *Proceedings of the Thiry-Fourth Annual ACM Symposium on Theory of Computing*, pages 741–750, 2002.

[7] L. Greengard and V. Rokhlin. A Fast Algorithm for Particle Simulations. *Journal of Computational Physics*, 73:325–248, 1987.

[8] A. G. Gray and A. W. Moore. '$N$-Body' Problems in Statistical Learning. In *NIPS*, volume 4, pages 521–527, 2000.

[9] A. G. Gray and A. W. Moore. Nonparametric Density Estimation: Toward Computational Tractability. In *SIAM International Conference on Data Mining*, 2003.

[10] D. Lee, A. G. Gray, and A. W. Moore. Dual-Tree Fast Gauss Transforms. In Y. Weiss, B. Schölkopf, and J. Platt, editors, *Advances in Neural Information Processing Systems 18*, pages 747–754. MIT Press, Cambridge, MA, 2006.

[11] D. Lee and A. G. Gray. Fast High-dimensional Kernel Summations Using the Monte Carlo Multipole Method. In *To appear in Advances in Neural Information Processing Systems 21*. 2009.

[12] S. Aluru, G. M. Prabhu, and J. Gustafson. Truly distribution-independent algorithms for the N-body problem. In *Proceedings of the 1994 conference on Supercomputing*, pages 420–428. IEEE Computer Society Press Los Alamitos, CA, USA, 1994.

[13] P. B. Callahan. *Dealing with Higher Dimensions: the Well-Separated Pair Decomposition and its applications*. PhD thesis, Johns Hopkins University, Baltimore, Maryland, 1995.

[14] P. B. Callahan and S. R. Kosaraju. A Decomposition of Multidimensional Point Sets with Applications to k-Nearest-Neighbors and n-body Potential Fields. *Journal of the ACM*, 62(1):67–90, January 1995.

[15] A. Beygelzimer, S. Kakade, and J.C. Langford. Cover trees for Nearest Neighbor. 2006. http://hunch.net/~jl/projects/cover_tree/paper/paper.ps.

[16] R. D. Skeel, I. Tezcan, and D. J. Hardy. Multiple Grid Methods for Classical Molecular Dynamics. *Journal of Computational Chemistry*, 23(6):673–684, 2002.

[17] A. G. Gray and A. W. Moore. Rapid Evaluation of Multiple Density Models. In *Artificial Intelligence and Statistics 2003*, 2003.

[18] A. G. Gray and A. W. Moore. Very Fast Multivariate Kernel Density Estimation via Computational Geometry. In *Joint Statistical Meeting 2003*, 2003. to be submitted to JASA.

[19] R. Krauthgamer and J. R. Lee. Navigating Nets: Simple Algorithms for Proximity Search. *15th Annual ACM-SIAM Symposium on Discrete Algorithms*, pages 791–801, 2004.

[20] K. Clarkson. Fast Algorithms for the All Nearest Neighbors Problem. In *Proceedings of the Twenty-fourth Annual IEEE Symposium on the Foundations of Computer Science*, pages 226–232, 1983.

